# Insights from Machine Learning Applied to Human Visual Classification

**Arnulf B. A. Graf and Felix A. Wichmann**
Max Planck Institute for Biological Cybernetics
Spemannstraße 38
72076 Tübingen, Germany
{arnulf.graf, felix.wichmann}@tuebingen.mpg.de

## Abstract

We attempt to understand visual classification in humans using both psychophysical and machine learning techniques. Frontal views of human faces were used for a gender classification task. Human subjects classified the faces and their gender judgment, reaction time and confidence rating were recorded. Several hyperplane learning algorithms were used on the same classification task using the Principal Components of the texture and shape representation of the faces. The classification performance of the learning algorithms was estimated using the face database with the true gender of the faces as labels, and also with the gender estimated by the subjects. We then correlated the human responses to the distance of the stimuli to the separating hyperplane of the learning algorithms. Our results suggest that human classification can be modeled by some hyperplane algorithms in the feature space we used. For classification, the brain needs more processing for stimuli close to that hyperplane than for those further away.

## 1 Introduction

The last decade has seen tremendous technological advances in neuroscience from the microscopic to the macroscopic scale (e.g. from multi-unit recordings to functional magnetic resonance imaging). On an algorithmic level, however, methods and understanding of brain processes are still limited. Here we report on a study combining psychophysical and machine learning techniques in order to improve our understanding of human classification of visual stimuli. What algorithms best describe the way the human brain classifies? Might humans use something akin to hyperplanes for classification? If so, is the learning rule as simple as in mean-of-class prototype learners or are more sophisticated algorithms better candidates?

In our experiments, subjects and machines classified human faces according to gender. The stimuli were presented and we collected the subjects' responses, which are the estimated gender, reaction time and confidence rating (sec.2). For every subject two personal new datasets were created: the original faces either with the true or with the subject's labels (true or estimated gender response). We then applied a Principal Component Analysis to a texture and shape representation of the faces. Various algorithms such as Support Vec-

tor Machines, Relevance Vector Machines, Prototype and K-means Learners (sec.3) were applied on this low-dimensional dataset with either the true or the subjects' labels. The resulting classification performances were compared, the corresponding decision hyperplanes were computed and the distances of the faces to the hyperplanes were correlated with the subjects' responses, the data being pooled among all subjects and stimuli or on a stimulus-by-stimulus basis (sec.4).

## 2 Human Classification Behaviour

We used grey-scale frontal views of human faces taken from the MPI face database [1]. Because of technical inhomogeneities of the faces in the database we post-processed each face such that all faces have same mean intensity, same pixel-surface area and are centred [2]. This processing stage is followed by a slight low-pass filtering of each face in the database in order to eliminate, as much as possible, scanning artifacts. The database is gender-balanced and contains 200 Caucasian faces (see Fig.1). Twenty-seven human

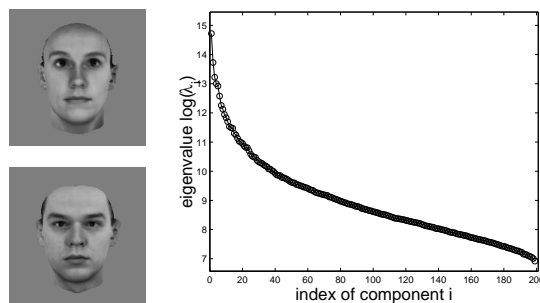

Figure 1: Female and male faces from the processed database (left). Eigenvalue spectrum from the PCA of our texture-shape representation (see sec.4): $\lambda_{min} = 1.01 \cdot 10^3$ (the last eigenvalue being 0 is not plotted) and $\lambda_{max} = 2.47 \cdot 10^6$ (right).

subjects were asked to classify the faces according to their gender and we recorded three responses: estimated class (i.e. female/male), reaction time (RT) and, after each estimated-class-response, a confidence rating (CR) on a scale from 1 (unsure) to 3 (sure). The stimuli were presented sequentially to the subjects on a carefully calibrated display using a modified Hanning window (a raised cosine function with a raising time of $t_{transient} = 500$ms and a plateau time of $t_{steady} = 1000$ms, for a total presentation time $t = 2000$ms per face). Subjects were asked to answer as fast as possible to obtain perceptual, rather than cognitive, judgements. Most of the time they responded well before the presentation of the stimulus had ended (mean RT over all stimuli and subjects was approximately 900ms). All subjects had normal or corrected-to-normal vision and were paid for their participation. Most of them were students from the University of Tübingen and all of them were naive to the purpose of the experiment.

Analysis of the classification performance of humans is based on signal detection theory [3] and we assume that, on the decision axis, the internal signal and noise distributions are Gaussian with same unit variance but different means. We define correct response probabilities for males (+) and females (−) as $P_+ = P(\hat{y} = 1|y = 1)$ and $P_- = P(\hat{y} = -1|y = -1)$ where $\hat{y}$ is the estimated class and $y$ the true class of the stimulus. The discriminability of both stimuli can then be computed as: $d' = Z(P_+) + Z(P_-)$ where $Z = \Phi^{-1}$, and $\Phi$ is the cumulative normal distribution with zero mean and unit variance. Averaged across subjects we obtain $d' = 2.85 \pm 0.73$. This value indicates that the classification task is comparatively easy for the subjects, although without being trivial (no ceiling effect). We observe a strong male bias (a large number of females

classified as males but very few males classified as females) and express this bias as: $\eta = Z^2(P_+) - Z^2(P_-) = 3.14 \pm 2.61$. The subplots of Fig.2 show the correlations of (a) RT and classification error, (b) classification error and CR, and (c) RT and CR. First,

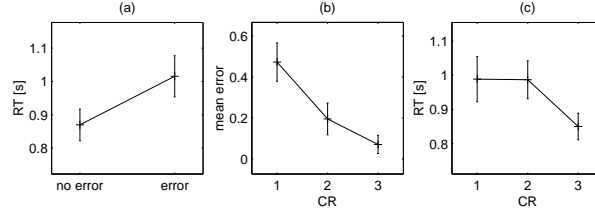

Figure 2: Human classification behaviour: mutual dependencies of the subject's responses.

RT's are longer for incorrect answers than for correct ones (a). Second, a high CR is correlated with a low classification error (b) and thus subjects have veridical knowledge about the difficulty of individual responses—this is certainly not the case in many low-level psychophysical settings. Third, the RT decreases as the CR increases (c), i.e. stimuli easy to classify are also classified rapidly. It may thus be concluded that a high error (or equivalently a low CR) implies higher RT's. This may suggest that patterns difficult to classify need more computation, i.e. longer processing, by the brain than patterns easy to classify.

## 3 Machine Learning Classifiers

In the following, various hyperplane classification algorithms are expressed as weighted dual space learners with different learning rules. Given a dataset $\{\vec{x}_i, y_i\}_{i=1}^p$, we assume classification is done in the input space, i.e. we consider linear kernels. Moreover, the input space is normalized since this has proved to be effective for some classifiers [4]. The hyperplanes can be written using a weight (or normal) vector $\vec{w}$ and an offset $b$ in order to yield a classification rule as $y(\vec{x}) = sign(\langle \vec{w} | \vec{x} \rangle + b)$ in the first three cases whereas in the last one, the decision rule is a collection of hyperplanes. These classifiers are compared on a two-dimensional toy dataset in Fig.3.

**Support Vector Machine** (SVM, [5]). The weight vector is given as: $\vec{w} = \sum_i \alpha_i y_i \vec{x}_i$ where $\vec{\alpha}$ is obtained by maximising $\sum_i \alpha_i - \frac{1}{2} \sum_{ij} y_i y_j \alpha_i \alpha_j \langle \vec{x}_i | \vec{x}_j \rangle$ subject to $\sum_i \alpha_i y_i = 0$ and $0 \leq \alpha_i \leq C$ where $C$ is a regularisation parameter, determined using for instance cross-validation. The offset is computed as: $b = \langle y_i - \langle \vec{w} | \vec{x}_i \rangle \rangle_{i|0<\alpha_i<C}$.

**Relevance Vector Machine** (RVM, [6]). The weight vector (incorporating here the offset) is expressed as $\vec{w} = \sum_i \alpha_i \vec{x}_i$. A Bernoulli distribution describes $P(\vec{y}|X, \vec{\alpha})$ where $X = \{\vec{x}_i\}_{i=1}^p$. A hyperparameter $\vec{\beta}$ is introduced in order to retrieve a sparse and smooth solution for $\vec{\alpha}$ using a Gaussian distribution for $P(\vec{\alpha}|\vec{\beta})$. Learning amounts to maximising $P(\vec{y}|X, \vec{\beta}) = \int P(\vec{y}|X, \vec{\alpha}) P(\vec{\alpha}|\vec{\beta}) d\vec{\alpha}$ with respect to $\vec{\beta}$. Since the latter is not integrable analytically, the Laplace approximation (local approximation of the integrand by a Gaussian) is used for resolution, yielding an iterative update scheme for $\vec{\beta}$.

**Prototype Learner** (Prot, [7]). Defining the prototypes $\vec{p}_\pm = \frac{\sum_{i=1}^p \vec{x}_i(y_i \pm 1)}{\sum_{i=1}^p (y_i \pm 1)} = \sum_{i|y_i=\pm 1} \alpha_i \vec{x}_i$ as the centre of mass of each class, the weight vector is then expressed as: $\vec{w} = \vec{p}_+ - \vec{p}_- = \sum_i \alpha_i y_i \vec{x}_i$ and the offset as: $b = \frac{\|\vec{p}_-\|^2 - \|\vec{p}_+\|^2}{2} = -\frac{\langle \vec{w} | \sum_i \alpha_i \vec{x}_i \rangle}{2}$.

**K-means Clustering with Nearest-neighbor Learner** (Kmean, [8]). Once the $K$ centres of the clusters for each class are computed using the K-means algorithm, one mean $\vec{k}_\pm(\vec{x}) = \sum_i \varphi_i^\pm(\vec{x}) \vec{x}_i$ for each class is selected for a pattern $\vec{x}$ using the nearest-neighbour rule. The weight is then computed as: $\vec{w}(\vec{x}) = \vec{k}_+(\vec{x}) - \vec{k}_-(\vec{x}) = \sum_i (\varphi_i^+(\vec{x}) - \varphi_i^-(\vec{x})) \vec{x}_i$,

the offset being given by: $b(\vec{x}) = \frac{\|\vec{k}_-(\vec{x})\|^2 - \|\vec{k}_+(\vec{x})\|^2}{2}$. Since the nearest-neighbour rule is used for each pattern, the decision function is *piecewise* linear. The appropriate value of $K$ is determined for instance using cross-validation.

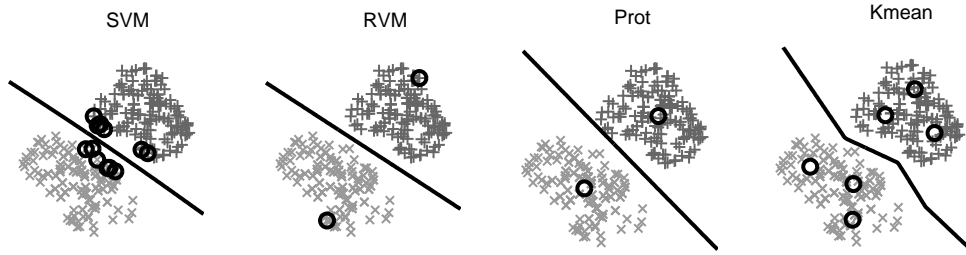

Figure 3: Two-dimensional toy example illustrating classification for a SVM, RVM, Prot and Kmean: the lines indicate the separating hyperplanes and the circles show the SVs, RVs, prototypes or means respectively.

## 4 Human Classification Behaviour Revisited by Machine

Each face taken from the MPI database is represented by three vectors: an intensity-standardised *texture* map, and space-standardised $x$- and $y$-flowfields representing the *shape*. The texture and shape vectors contain the information required to generate a specific face from an "average" reference face by putting each face of the database into *correspondence*. This format makes intensity and structural information about the faces explicit. For the sake of numerical tractability, especially when using cross-validation methods, the dimension of the image vectors has to be reduced to be usable by machine learning algorithms. We use Principal Component Analysis (PCA) to represent the concatenated texture- and shape vectors of each face of size $3 \cdot 256^2$ in only 200 dimensions. In contrast to [9] where PCA is applied only to the intensity (or pixel) information of standard images, the use of PCA on the texture-shape representation forces learning machines to encode information about local structure and spatial correspondences.

It may be argued that the Principal Components of faces form a biologically-plausible basis for representation of faces [10], the so-called eigenfaces. Standard PCA on the images themselves may thus be considered a biologically-plausible representation of faces. Given that we use PCA on texture and shape, any claim of biological plausibility of our representation is somewhat tenuous, however.

The variant of PCA considered in this paper searches to express the eigenvectors as linear combinations of the data vectors [10, 11]. It has the computational advantage over classic PCA that it does not require the computation of a correlation between the dimensions of the input but between the patterns of the input. For the stimuli considered here, the eigenvalue spectrum as shown in Fig.1 is a monotonically decreasing function with no flat regions. Thus PCA seems to be a sensible choice to represent the human face stimuli used in this study (for a comparative study of PCA against Locally Linear Embedding, where PCA is clearly superior for machine learning purposes, see [2]).

### 4.1 Classification Performance of Man and Machine
We compare the classification performance of man and machine in plot (a) of Fig.4. For humans, the classification error on the true dataset is obtained by comparing the estimated gender (class) to the true one. The classification error on the subject dataset, seen as a measure for the mean consistency between subjects, is the mean over all subjects of the

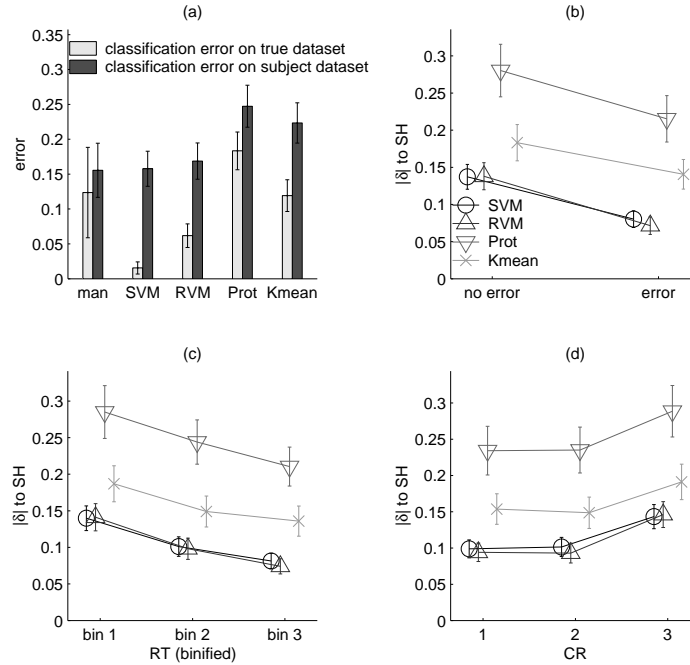

Figure 4: Classification performance of man and machine on the true and subject datasets (a) and correlation of behaviour of man (classification error, RT and CR) with machine ($|\delta|$) for data pooled across subjects and stimuli (b-d).

mean classification error the other subjects made on the stimuli presented to each subject by defining as an error when the other subjects responded differently than the considered subject. For machines the classification error is obtained for the dataset with either the true or the subject's labels by using a single 10-fold cross-validation for RVM and Prot and a double 10-fold cross-validation to determine $C$ for SVMs and $K$ for Kmeans. Since every subject gets a different set of $148$ randomly chosen faces from the $200$ available, the mean and standard error of the classification errors of man or machine for each dataset is plotted.

When classifying the dataset with the true labels, the combination of PCA with Kmean yields a classification performance comparable to that of humans. The better classification performance of Kmean compared to the simple prototype classifier may be explained by the piecewise linear decision function. The prototype classifier, popular in neuroscience, psychology and philosophy, performs on average worse than humans. Either humans do not classify gender using prototypes in the linear PCA space, or they use prototypes but not the PCA representation, or, of course, they use neither.

An intriguing fact is that SVMs and RVMs perform better than man, which is contrary to what is reported in [5, 12] where human experts and machines are tested on digits from the postal service database USPS. The context of the study presented here is different, however. Our subjects were presented with human faces with some high-level features such as hair, beards, or glasses removed. However, such features were likely used by the subjects to create their representation of gender-space during their lifetime. Subjects are thus trained on one type of data and tested on another. The machines on the other hand are trained and tested on the same type of stimuli. This may explain the quite disappointing performance of man in such a biologically-relevant task compared to machine.

However while humans learn the gender classification during their lifetime, it seems that they solve the problem in a manner not as optimal from a statistical point of view as SVMs or RVMs, but similarly to Kmeans and better than prototype learners.

The classification on the subject's labels represents the ability of the classifier to learn what we, based on the responses of the subjects, presume to be their internal representation of face-space. The machines have more difficulty in learning the dataset with the subject's labels than the one with the true labels. Given our aim of re-creating the subjects' decision boundaries using artificial classifiers—to compare human response patterns to machine learning concepts—this makes SVM and RVM good, Kmean a mediocre, and the prototype learner a rather poor candidate for this enterprise using the PCA representation.

### 4.2 Correlation of Behaviour of Man with Machine

Here we correlate the classification behaviours of man and machine. The results are summarized in plot (b-d) of Fig.4 and in Fig.5 where the parameters are averaged over the subjects as before. This type of data analysis simply correlates the subject's classification error, RT, and CR to the distance $|\delta(\vec{x}_i)| = \frac{|\langle\vec{w}|\vec{x}_i\rangle + b|}{\|\vec{w}\|}$ of the face stimuli to the separating hyperplane (SH) obtained for the four types of hyperplane classifiers (in the case of Kmean this distance is computed for each pattern with respect to the SH constructed using its nearest mean of each class.). The hyperplanes are determined using cross-validation (see above) on the dataset with the subject's labels. The distance of a pattern $\vec{x}$ to the SH is then calculated using the hyperplane computed using the training set corresponding to the testing set $\vec{x}$ is belonging to. Notice that $|\delta|$ reflects the construction rule of the classification hyperplane rather than the generalisation ability of the algorithm. SVMs maximise the distance to the nearest point but not the average distance to all points, which may yield a small value of $|\delta|$. Moreover the number of SVs, here $\sharp(SV) = 74 \pm 1$ out of 148 patterns, indicates that most patterns are close to the SH since classification is done in a space of dimensionality 200. The number of RVs, $\sharp(RV) = 9 \pm 0$, is comparatively small, this sparsity being a well-known feature of RVMs.

Looking at Fig.4 (b-d) where the data is averaged across subjects and stimuli, we observe, first, that the error of the subjects is high for $|\delta|$ low, suggesting that elements near the SH are more difficult to classify. Second $|\delta|$ is low for high RT's: the elements near the SH seem to require more processing in the brain resulting in a higher RT. Third, the high CR for high $|\delta|$ indicates that the subjects are sure when stimuli are far from SH. Thus elements far from the SH are classified more accurately, faster and with higher confidence than those near to the SH. In order to compare the classifiers, we proceed as below.

Thus far we only considered data averaged across all face-stimuli. In the following we assess the relation between the distance of each face representation to the SH and the mean across all subjects of one of their responses (classification error, RT or CR) for that face. We perform a non-parametric rank correlation analysis using the tied rank of the subject's response and of $|\delta|$ across the set of 200 faces. Fig.5 presents the resulting scatter plots for each classifier and for each type of response. Qualitatively, it seems that RVMs show most and prototype learners least correlation between the subject's response and $|\delta|$. In order to compare these behaviours in a more quantitative manner, we indicate in fig.5 Spearman's rank correlation coefficients $r$ (linear correlation between the tied rank of one variable and the tied rank of the other) between the parameter of machine (distance of a face to the SH) and the responses of man (classification error, RT and CR). Under the null hypothesis of no correlation between man and machine, the variable $z = r\sqrt{N-1}$ follows a standard normal distribution, $N = 200$ being the number of points in the scatter plots, and the significance of the hypothesis test is computed as $P = \Phi(z)$ where $\Phi$ is the cumulative normal distribution with zero mean and unit variance. We get for all cases $P < 5 \cdot 10^{-4}$ which allows us to reject the null hypothesis with a high degree of confidence.

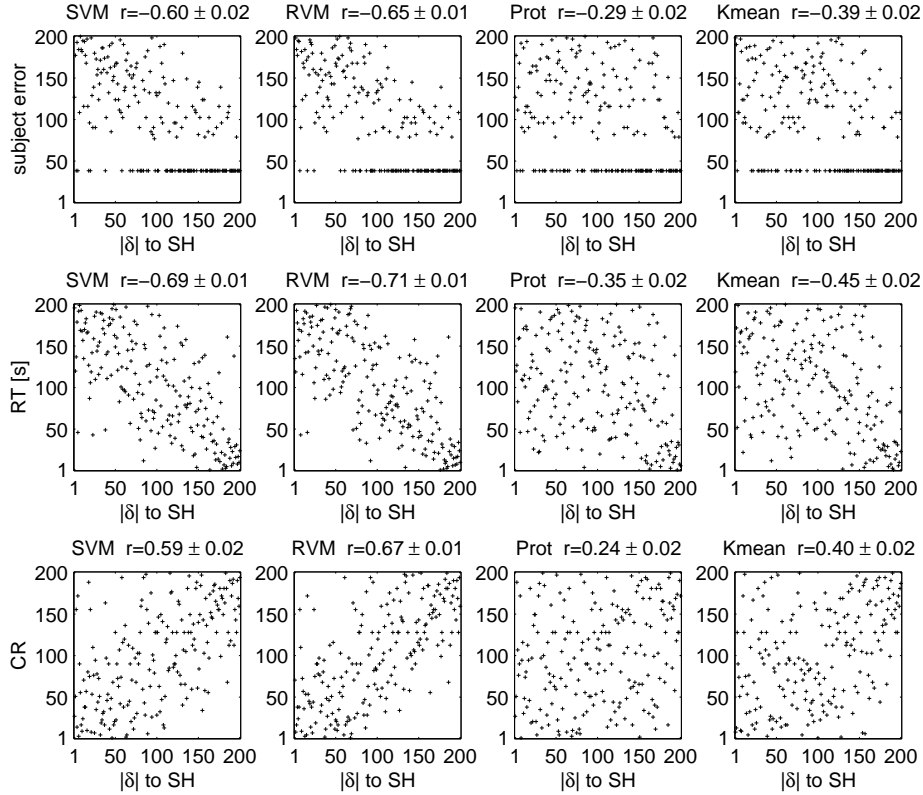

Figure 5: Scatter plots relating the subjects' responses (classification error, RT and CR) to the distance $|\delta|$ to the SH for each face in the database, the pooling being done across subjects.

From these results it can be seen that RVMs correlate best all the subject's responses with the distances of the stimuli to the SH. The RT seems to be the performance measure where most correlation between man and machine can be asserted although all performance measures are related as shown in sec.2. The prototype algorithm again behaves in the least human-like manner of the four classifiers. The correlation between the classification behaviour of man and machine indicates for RVMs, and to some extent SVMs, that heads far from the SH are more easily processed by humans. It may be concluded that the brain needs to do more processing (higher RT) to classify stimuli close to the decision hyperplane, while stimuli far from it are classified more accurately (low error) and with higher confidence (high CR). Human classification behaviour can thus be modeled by hyperplane algorithms; a piecewise linear decision function as found in Kmean seems however to be not biologically-plausible.

## 5   Conclusions

Our study compared classification of faces by man and machine. Psychophysically we noted that a high classification error and a low CR for humans is accompanied by a longer processing of information by the brain (a longer RT). Moreover, elements far from the SH are classified more accurately, faster and with higher confidence than those near to the SH. We also find three noteworthy results. First, SVMs and RVMs can learn to classify faces

using the subjects' labels but perform much better when using the true labels. Second, correlating the average response of humans (classification error, RT or CR) with the distance to the SH on a face-by-face basis using Spearman's rank correlation coefficients shows that RVMs recreate human performance most closely in every respect. Third, the mean-of-class prototype, its popularity in neuroscience notwithstanding, is the least human-like classifier in all cases examined.

Obviously our results rely on a number of crucial assumptions: first, all measurements were done in a linear space; second, the conclusions are only valid given the PCA representation (pre-processing). Third, when rejecting the prototype learner as a plausible candidate for human classification we assume the representativeness of our face space: we assume that the mean face of our human subjects' is close to the sample mean of our database. Clearly, a larger face database would be welcome, but is not trivial as we need texture maps and the corresponding shapes. Finally, there is the different learning regime. Machines were trained on the dataset proper, whereas humans were assumed to have extracted the relevant information during their lifetime, and they were tested on faces with some cues removed. However, the representation we used does allow the genders to be separated well, as shown by the SVM classification performance on the true labels. As a first attempt to extend the neuroscience community's toolbox with machine learning methods we believe to have shown the fruitfulness of this approach.

### Acknowledgements

The authors would like to thank Volker Blanz for providing the face database and the flow-field algorithms. In addition we are grateful to Gökhan Bakır, Heinrich Bülthoff, Jez Hill, Carl Rasmussen, Gunnar Rätsch, Bernhard Schölkopf and Vladimir Vapnik for helpful comments and suggestions. AG was supported by a grant from the European Union (IST 2000-29375 COGVIS).

## References

[1] V. Blanz and T. Vetter. A Morphable Model for the Synthesis of 3D Faces. Proc. Siggraph99, pp. 187-194. Los Angeles: ACM Press, 1999.

[2] A. B. A. Graf and F. A. Wichmann. Gender Classification of Human Faces. Proceedings of the BMCV, Springer LNCS 2525, 491-501, 2002.

[3] T. D. Wickens. *Elementary Signal Detection Theory*. Oxford University Press, 2002.

[4] A. B. A. Graf, A. J. Smola, and S. Borer. Classification in a Normalized Feature Space using Support Vector Machines. IEEE Transactions on Neural Networks 14(3), 597-605, 2003.

[5] V. N. Vapnik. *The Nature of Statistical Learning Theory*. Springer, 1995.

[6] M. E. Tipping. Sparse Bayesian learning and the relevance vector machine. Journal of Machine Learning Research 1, 211-244, 2001.

[7] S. K. Reed. Pattern Recognition and Categorization. Cognitive Psychology 3, 382-407, 1972.

[8] R. O. Duda, P.E. Hart, and D.G. Stork. *Pattern Classification*. John Wiley & Sons, 2001.

[9] L. Sirovich and M. Kirby. Low-Dimensional Procedure for the Characterization of Human Faces. Journal of the Optical Society of America A, 4(3), 519-524, 1987.

[10] M. Turk and A. Pentland. Eigenfaces for Recognition. Journal of Cognitive Neuroscience, 3(1), 1991.

[11] B. Schölkopf, A. Smola, and K.-R. Müller. Nonlinear Component Analysis as a Kernel Eigenvalue Problem. Neural Computation, 10, 1299-1319, 1998.

[12] J. Bromley and E. Säckinger. Neural-network and K-nearest-neighbor Classifiers. Technical Report 11359-910819-16TM, AT&T, 1991.
